# Who's in the Picture?

**Tamara L. Berg, Alexander C. Berg, Jaety Edwards and D.A. Forsyth**
Computer Science Division
U.C. Berkeley
Berkeley, CA 94720
millert@cs.berkeley.edu

## Abstract

The context in which a name appears in a caption provides powerful cues as to who is depicted in the associated image. We obtain 44,773 face images, using a face detector, from approximately half a million captioned news images and automatically link names, obtained using a named entity recognizer, with these faces. A simple clustering method can produce fair results. We improve these results significantly by combining the clustering process with a model of the probability that an individual is depicted given its context. Once the labeling procedure is over, we have an accurately labeled set of faces, an appearance model for each individual depicted, and a natural language model that can produce accurate results on captions in isolation.

## 1 Introduction

It is a remarkable fact that pictures and their associated annotations are complementary. This observation has been used to browse museum collections ([1]) and organize large image collections ([2, 7, 12, 13]). All of these papers use fairly crude "bag of words" models, treating words as floating tags and looking at the co-occurrence of image regions and annotated words. In this paper, we show that significant gains are available by treating language more carefully.

Our domain is a large dataset of news photographs with associated captions. A face detector is used to identify potential faces and a named entity recognizer to identify potential names. Multiple faces and names from one image-caption pair are quite common. The problem is to find a correspondence between some of the faces and names. As part of the solution we learn an appearance model for each pictured name and the likelihood of a particular instance of a name being pictured based on the surrounding words and punctuation.

**Face recognition** cannot be surveyed reasonably in the space available. Reviews appear in [6, 10, 11]. Although face recognition is well studied, it does not work very well in practice [15]. One motivation for our work is to take the large collection of news images and captions as semi-supervised input and and produce a fully supervised dataset of faces labeled with names. The resulting dataset exhibits many of the confounding factors that make real-world face recognition difficult, in particular modes of variation that are not found in face recognition datasets collected in laboratories. It is important to note that this task is easier than general face recognition because each face has only a few associated names.

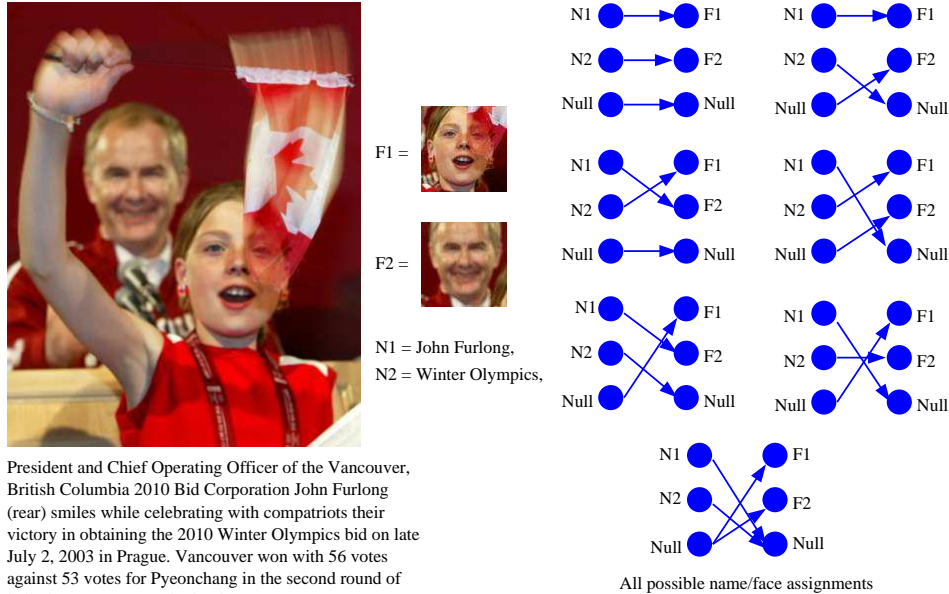

F1 =

F2 =

N1 = John Furlong,
N2 = Winter Olympics,

President and Chief Operating Officer of the Vancouver,
British Columbia 2010 Bid Corporation John Furlong
(rear) smiles while celebrating with compatriots their
victory in obtaining the 2010 Winter Olympics bid on late
July 2, 2003 in Prague. Vancouver won with 56 votes
against 53 votes for Pyeonchang in the second round of
balloting at an IOC gathering in Prague. REUTERS/Petr
Josek

All possible name/face assignments

Figure 1: **Left:** *A typical data item consisting of a picture and its associated caption.*
**Center:** *Detected faces and names for this data item.* **Right:** *The set of possible corre-*
*spondences for this data item. Our model allows each face to be assigned to at most one*
*name, each name to be assigned to at most one face, and any face or name to be assigned*
*to Null. Our named entity recognizer occasionally identifies strings that do not refer to*
*actual people (e.g. "Winter Olympics"). These names are assigned low probability under*
*our model and therefore their assignment to a face is unlikely. EM iterates between com-*
*puting the expectation of the possible face-name correspondences and updating the face*
*clusters and language model. Unusually, we can afford to compute all possible face-name*
*correspondences for a data item since the number of possibilities is small. For this item,*
*we correctly choose the best matching "F1 to Null", "N2 to Null", and "F2 to N1".*

**Language:** Quite simple common phenomena in captions suggest using a language model.
First, our named entity recognizer occasionally marks incorrect names like "United Na-
tions". The context in which these incorrect detections occur suggest that they do not refer
to actual people. Second, name-context pairs can be weighted according to their probabil-
ity. In a caption such as "Michael Jackson responds to questioning Thursday, Nov. 14, 2002
in Santa Maria Superior Court in San ta Maria, Calif., during a $21 million lawsuit brought
against him by Marcel Avram for failing to appear at two millennium concerts...", Michael
Jackson appears in a more favorable context (at the beginning of the caption, followed by
a verb) than Marcel Avram (near the middle of the caption, followed by a preposition).

**Our approach** combines a simple appearance model using kPCA and LDA, with a lan-
guage model, based on context. We evaluate both an EM and maximum likelihood clus-
tering and show that incorporating language with appearance produces better results than
using appearance alone. We also show the results of the learned natural language classifier
applied to a set of captions in isolation.

## 2    Linking a face and language model with EM

A natural way of thinking about name assignment is as a hidden variable problem where
the hidden variables are the correct name-face correspondences for each picture. This sug-
gests using an expectation maximization (EM) procedure. EM iterates between computing

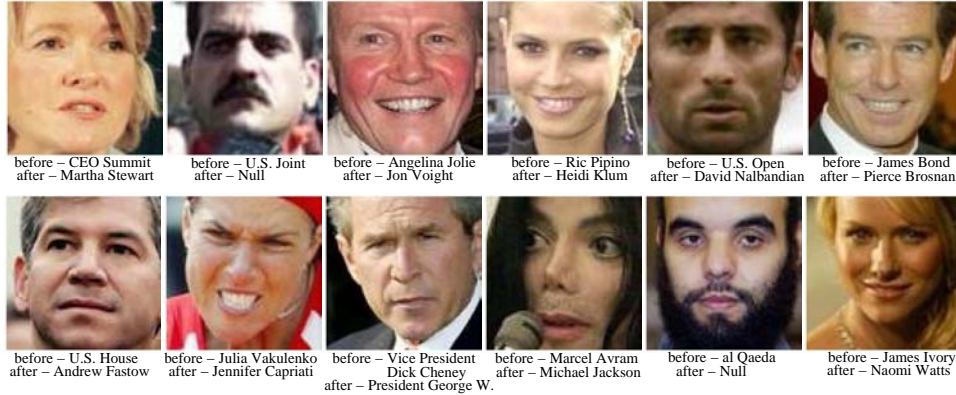

before – CEO Summit
after – Martha Stewart

before – U.S. Joint
after – Null

before – Angelina Jolie
after – Jon Voight

before – Ric Pipino
after – Heidi Klum

before – U.S. Open
after – David Nalbandian

before – James Bond
after – Pierce Brosnan

before – U.S. House
after – Andrew Fastow

before – Julia Vakulenko
after – Jennifer Capriati

before – Vice President
Dick Cheney
after – President George W.

before – Marcel Avram
after – Michael Jackson

before – al Qaeda
after – Null

before – James Ivory
after – Naomi Watts

Figure 2: *Names assigned using our raw clustering procedure* **(before)** *and incorporating a language model* **(after)**. *Our named entity recognizer occasionally detects incorrect names (e.g. "CEO Summit"), but based on context the language model assigns low probabilities to these names, making their assignment unlikely. When multiple names are detected like "Julia Vakulenko" and "Jennifer Capriati", the probability for each name depends on its context. The caption for this picture reads "American Jennifer Capriati returns the ball to her Ukrainian opponent Julia Vakulenko in Paris during..." The language model prefers to assign the name "Jennifer Capriati" because its context (beginning of the caption followed by a present tense verb) indicates it is more likely to be pictured than "Julia Vakulenko" (middle of the caption followed by a preposition). For pictures such the man labeled "al Qaeda" to "Null" where the individual is not named in the caption, the language model correctly assigns "Null" to the face. As table 1 shows, incorporating a language model improves our face clusters significantly.*

the expected values of the set of face-name correspondences (given a face clustering and language model) and updating the face clusters and language model given the correspondences. Unusually, it is affordable to compute the expected value of all possible face-name correspondences for a data item since the number of possibilities is small.

To use EM we need a model of how pictures are generated. Generative model:

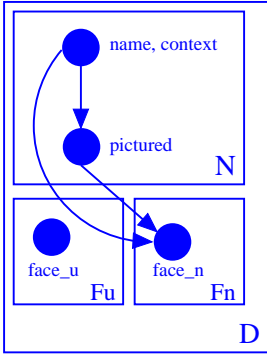

To generate a data item:

1. Choose N, the number of names, and F, the number of faces.
2. Generate N *name, context* pairs.
3. For each of these *name, context* pairs, generate a binary variable $pictured$ conditioned on the context alone (from $P(pictured|context)$).
4. For each $pictured = 1$, generate a face from $P(face|name)$ ($Fn = \sum pictured$).
5. Generate $Fu = F - Fn$ other faces from $P(face)$.

The parameters necessary for the EM process are $P(face|name)$ (sec 2.2), the probability that a face is generated by a given name, $P(pictured|context)$ (sec 2.3), the probability that a name is pictured given its context, and $P(face)$ the probability that a face is generated without a name.

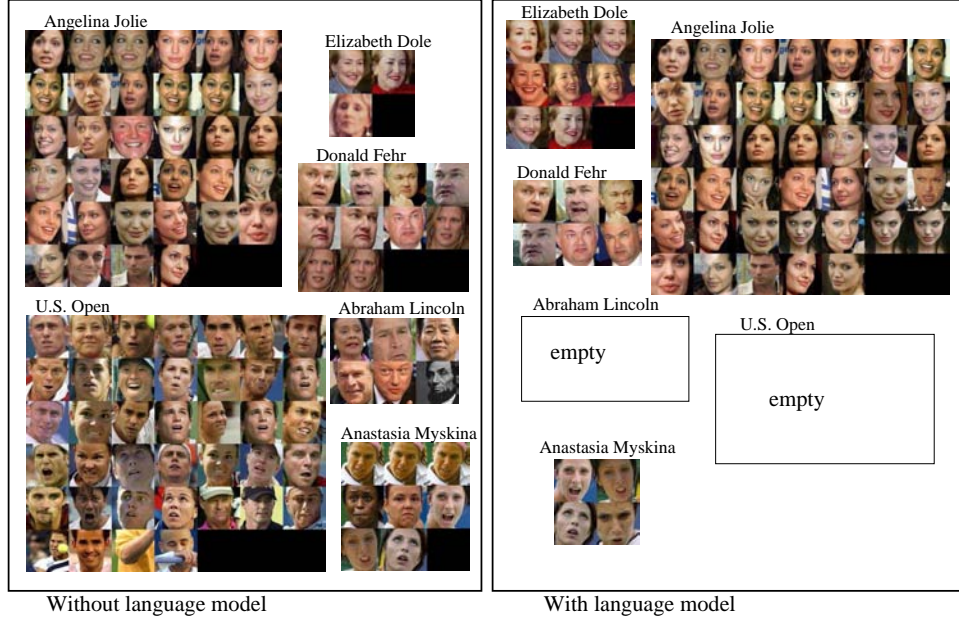

Without language model        With language model

Figure 3: **Left:** *Example clusters using only appearance to cluster.* **Right:** *The same clusters, but using appearance + language to cluster. Some clusters get larger (Elizabeth Dole, Angelina Jolie) because of the inclusion of more correct faces. Some clusters get smaller (Anastasia Myskina) because of the exclusion of incorrect faces. All clusters get more accurate because the language model is breaking ambiguities and giving the clustering a push in the right direction. Some clusters that do not refer to actual people, "U.S. Open" completely disappear using the language model. Other clusters like "Abraham Lincoln" (who is a person, but whose associated pictures most often portray people other than "Abraham Lincoln") become empty when using the language model, presumably because these faces are assigned to the correct names.*

## 2.1 Name Assignment

For each image-caption pair, we calculate the costs of all possible assignments of names to faces (dependent upon the associated faces and names) and use the best such assignment. An example of the extracted names, faces and all possible assignments can be seen in figure 1.

The likelihood of picture $x_i$ under assignment $a_j$, of names to faces under our generative model is:

$$L(x_i, a_j) = P(N)P(F)P(n_1, c_1)...P(n_n, c_n)*$$
$$\prod_\alpha P(pictured_\alpha | c_\alpha) P(f_{\sigma(\alpha)} | n_\alpha) \prod_\beta (1 - P(pictured_\beta | c_\beta)) \prod_\gamma P(f_\gamma)$$

Where P(N) is the probability of generating N names, P(F) is the probability of generating F faces, and $P(n_i, c_i)$ are the probabilities of generating $name_i$ and context $c_i$. In assignment $a_j$, $\alpha$ indexes into the names that are pictured, $\sigma(\alpha)$ indexes into the faces assigned to the pictured names, $\beta$ indexes into the names that are not pictured and $\gamma$ indexes into the faces without assigned names. The terms $P(N)P(F)P(n_1, c_1)...P(n_n, c_n)$ are not dependent on the assignment so we can ignore them when calculating the probability of an assignment and focus on the remaining terms.

| |
|---|
| **IN Pete Sampras IN** of the U.S. celebrates his victory over Denmark's **OUT Kristian Pless OUT** at the **OUT U.S. Open OUT** at Flushing Meadows August 30, 2002. Sampras won the match 6-3 7- 5 6-4. REUTERS/Kevin Lamarque |
| Germany's **IN Chancellor Gerhard Schroeder IN**, left, in discussion with France's **IN President Jacques Chirac IN** on the second day of the EU summit at the European Council headquarters in Brussels, Friday Oct. 25, 2002. EU leaders are to close a deal Friday on finalizing entry talks with 10 candidate countries after a surprise breakthrough agreement on Thursday between France and Germany regarding farm spending.(AP Photo/European Commission/HO) |
| 'The Right Stuff' cast members **IN Pamela Reed IN**, (L) poses with fellow cast member **IN Veronica Cartwright IN** at the 20th anniversary of the film in Hollywood, June 9, 2003. The women played wives of astronauts in the film about early United States test pilots and the space program. The film directed by **OUT Philip Kaufman OUT**, is celebrating its 20th anniversary and is being released on DVD. REUTERS/Fred Prouser |
| Kraft Foods Inc., the largest U.S. food company, on July 1, 2003 said it would take steps, like capping portion sizes and providing more nutrition information, as it and other companies face growing concern and even lawsuits due to rising obesity rates. In May of this year, San Francisco attorney **OUT Stephen Joseph OUT**, shown above, sought to ban Oreo cookies in California – a suit that was withdrawn less than two weeks later. Photo by Tim Wimborne/Reuters REUTERS/Tim Wimborne |

Figure 4: *Our new procedure gives us not only better clustering results, but also a natural language classifier which can be tested on captions in isolation.* **Above:** *a few captions labeled with IN (pictured) and OUT (not pictured) using our learned language model. Our language model has learned which contexts have high probability of referring to pictured individuals and which contexts have low probabilities. We observe an 85% accuracy of labeling who is portrayed in a picture using only our language model. The top 3 labelings are all correct. The last incorrectly labels "Stephen Joseph" as not pictured when in fact he is the subject of the picture. Some contexts that are often incorrectly labeled are those where the name appears near the end of the caption (usually a cue that the individual named is not pictured). Some cues we could add that should improve the accuracy of our language model are the nearness of words like "shown", "pictured", or "photographed".*

The complete data log likelihood is:

$$\sum_{i\epsilon pics} \left[ \sum_{j\epsilon ass} (P_{ij} log(L(x_i, a_j))) \right]$$

Where $P_{ij}$ is an indicator variable telling which correspondence occurred in this data item. The $P_{ij}$ are missing data whose expectations are computed in the E step.

This gives a straightforward EM procedure:

- E – update the $P_{ij}$ according to the normalized probability of picture i with assignment j.
- M – maximize the parameters $P(face|name)$ and $P(pictured|context)$ using soft counts.

### 2.2 Modeling the appearance of faces – $P(face|name)$

We model appearance using a mixture model with one mixture element per name in our lexicon. We need a representation for faces in a feature space where comparisons are helpful. Our representation is obtained by rectification of the faces followed by kernel principal components analysis (kPCA) and linear discriminant analysis (LDA) (details in [5]). We model the distributions $P(face|name)$ using gaussians with fixed covariance.

To obtain features we first automatically rectify all faces to a canonical pose. Five support vector machines are trained as feature detectors (corners of the left and right eyes, corners

| Model | EM | MM |
|---|---|---|
| Appearance Model, No Lang Model | 56% | 67% |
| Appearance Model + Lang Model | 72% | 77% |

Table 1: **Above:** *We randomly selected a set of 1000 faces from our dataset and hand labeled them with their correct names. Here we show what percentage of those faces are correctly labeled by each of our methods (EM and maximal correspondence clustering, MM). For both methods, incorporating a language model improves their respective clusterings greatly. Standard statistical knowledge says that using the expected values should perform better than simply choosing the maximal assignment at each step (MM). However, we have found that using the maximal assignment works better than taking an expectation. One reason this could be true is that EM averages incorrect faces into the appearance model, making the mean unstable.*

of the mouth and the tip of the nose) using features consisting of the geometric blur of [4] applied to grayscale patches. We then use kPCA ([16]) to reduce the dimensionality of our data and compute linear discriminants ([3]) on the single name, single face pictures.

Because of the large size of our dataset, we cannot compute the kernel matrix, K for kPCA, directly. Instead we use an approximation to calculate the eigenvectors of K, the Nyström approximation (cf [17, 9]). The Nyström approximation computes two exact subsets of K and uses these to efficiently approximate the rest of K and its eigenvectors (details in [5]).

### 2.3  Language Model – $P(pictured|context)$

Our language model assigns a probability to each name based on its context within the caption. These distributions, $P(pictured|context)$, are learned using counts of how often each context appears describing an assigned name, versus how often that context appears describing an unassigned name. We have one distribution for each possible context cue, and assume that context cues are modeled independently (because we lack enough data to model them jointly).

For context, we use a variety of cues: the part of speech tags of the word immediately prior to the name and immediately after the name within the caption (modeled jointly), the location of the name in the caption, and the distances to the nearest ",", ".", "(", ")", "(L)", "(R)" and "(C)". We tried adding a variety of other language model cues, but found that they did not increase the assignment accuracy.

The probability of being pictured given multiple context cues (where $C_i$ are the different independent context cues) can be formed using Bayes rule:

$$P(pictured|C_1, C_2, ...C_n) = \frac{P(pictured|C_1)...P(pictured|C_n)}{P(pictured)^{n-1}}$$

We compute maximum likelihood estimates of each of $P(pictured|C_i)$ and $P(pictured)$ using soft counts.

### 2.4  Best correspondence and mean correspondence

Given our hidden variable problem of determining correct name-face assignments, from a statistics point of view EM seems like the most favorable choice. However, many computer vision problems have observed better results by choosing maximum over expected values. We have tried both methods and found that using the maximal assignment produced better results (table 1). One reason this might be true is that for cases where there is a clear best assignment the max and the average are basically equivalent. For cases where there is no clear best, EM averages over assignments, producing a mean that has no real meaning since it is an average of different people's faces.

| Classifier | labels correct | IN correct | OUT correct |
|---|---|---|---|
| Baseline | 67% | 100% | 0% |
| EM Labeling with Language Model | 76% | 95% | 56% |
| MM Labeling with Language Model | 84% | 87% | 76% |

Table 2: **Above:** *Results of applying our learned language model to a test set of 430 captions (text alone). In our test set, we hand labeled each detected name with IN/OUT based on whether the referred name was pictured within the corresponding picture. We then tested how well our language model could predict those labels ("labels correct" refers to the total percentage of names that were correctly labeled, "IN correct" the percentage of pictured names correctly labeled, and "OUT correct" the percentage of not pictured names correctly labeled). The baseline figure gives the accuracy of labeling all names as pictured. Using EM to learn a language model gives an accuracy of 76% while using a maximum likelihood clustering gives 84%. Again the maximum likelihood clustering outperforms EM. Names that are most often mislabeled are those that appear near the end of the caption or in other contexts that usually denote a name being not pictured.*

The Maximal Assignment process is nearly the same as the EM process except instead of calculating the expected value of each assignment only the maximal assignment is nonzero.

The Maximal Assignment procedure:
- M1 – set the maximal $P_{ij}$ to 1 and all others to 0.
- M2 – maximize the parameters $P(face|name)$ and $P(pictured|context)$ using counts.

## 3   Results

We have collected a dataset consisting of approximately half a million news pictures and captions from Yahoo News over a period of roughly two years.

**Faces:** Using the face detector of [14], we extract 44,773 large well detected face images. Since these pictures were taken "in the wild" rather than under fixed laboratory conditions, they represent a broad range of individuals, pose, expression, illumination conditions and time frames. Our face recognition dataset is more varied than any other to date.

**Names:** We use an open source named entity recognizer ([8]) to detect proper names in each of the associated captions. This gives us a set of names associated with each picture.

**Scale:** We obtain 44,773 large and reliable face detector responses. We reject face images that cannot be rectified satisfactorily, leaving 34,623. Finally, we concentrate on images within whose captions we detect proper names, leaving 30,281, the final set we cluster on.

### 3.1   Quantitative Results

Incorporating a natural language model into face clustering produces much better results than clustering on appearance alone. As can be seen in table 1, using, only appearance produces an accuracy of 67% while appearance + language gives 77%. For face labeling, using the maximum likelihood assignment (MM) rather than the average (EM) produces better results (77% vs 72%).

One neat by-product of our clustering is a natural language classifier. We can evaluate that classifier on text without associated pictures. In table 2, we show results for labeling names with pictured and not pictured using our language model. Using the language model we correctly label 84% of the names while the baseline (labeling everyone as pictured) only gives 67%. The maximum likelihood assignment also produces a better language model than EM (76% vs 84%). A few things that our language model learns as indicative of being pictured are being near the beginning of the caption, being followed by a present tense verb, and being near "(L)", "(R)", or "(C)".

## 4 Discussion

We have shown previously ([5]) that a good clustering can be created using names and faces. In this work, we show that by analyzing language more carefully we can produce a much better clustering (table 1). Not only do we produce better face clusters, but we also learn a natural language classifier that can be used to determine who is pictured from text alone (table 2). We have coupled language and images, using language to learn about images and images to learn about language.

The next step will be to try to learn a language model for free text on a webpage. One area we would like to apply this to is improving google image search results. Using a simple image representation and a modified context model perhaps we could link google images with the words on the surrounding webpages to improve search results.

## References

[1] K. Barnard, D.A. Forsyth, "Clustering Art," *Computer Vision and Pattern Recognition*, 2001

[2] K. Barnard, P. Duygulu, N. de Freitas, D.A. Forsyth, D. Blei, and M.I. Jordan, "Matching Words and Pictures," *Journal of Machine Learning Research*, Vol 3, pp 1107-1135, 2003.

[3] P. Belhumeur, J. Hespanha, D. Kriegman "Eigenfaces vs. Fisherfaces: Recognition Using Class Specific Linear Projection" *Transactions on Pattern Analysis and Machine Intelligence*, Special issue on face recognition, pp. 711-720, July 1997.

[4] A.C. Berg, J. Malik, "Geometric Blur for Template Matching," *Computer Vision and Pattern Recognition*,Vol I, pp. 607-614, 2001.

[5] T.L. Berg, A.C. Berg, J. Edwards, M. Maire, R. White, E. Learned-Miller, D.A. Forsyth "Names and Faces in the News" *Computer Vision and Pattern Recognition*, 2004.

[6] V. Blanz, T. Vetter, "Face Recognition Based on Fitting a 3D Morphable Model," *Transactions on Pattern Analysis and Machine Intelligence* Vol. 25 no.9, 2003.

[7] C. Carson, S. Belongie, H. Greenspan, J. Malik, "Blobworld – Image segmentation using expectationmaximization and its application to image querying," *IEEE Transactions on Pattern Analysis and Machine Intelligence*, 24(8), pp. 1026–1038, 2002.

[8] H. Cunningham, D. Maynard, K. Bontcheva, V. Tablan, "GATE: A Framework and Graphical Development Environment for Robust NLP Tools and Applications," *40th Anniversary Meeting of the Association for Computational Linguistics"*, Philadelphia, July 2002.

[9] C. Fowlkes, S. Belongie, F. Chung, J. Malik, "Spectral Grouping Using The Nyström Method," *TPAMI*, Vol. 26, No. 2, February 2004.

[10] R. Gross, J. Shi and J. Cohn, "Quo Vadis Face Recognition?," *Third Workshop on Empirical Evaluation Methods in Computer Vision*, December, 2001.

[11] R. Gross, I. Matthews, and S. Baker, "Appearance-Based Face Recognition and Light-Fields," *Transactions on Pattern Analysis and Machine Intelligence*, 2004.

[12] V. Lavrenko, R. Manmatha., J. Jeon, "A Model for Learning the Semantics of Pictures," *Neural Information Processing Systems*, 2003

[13] J. Li and J. Z. Wang, "Automatic Linguistic Indexing of Pictures by a Statistical Modeling Approach," *Transactions on Pattern Analysis and Machine Intelligence*, vol. 25, no. 9, pp. 1075-1088, 2003

[14] K. Mikolajczyk "Face detector," *Ph.D report*, INRIA Rhone-Alpes

[15] J. Scheeres, "Airport face scanner failed", *Wired News*, 2002. http://www.wired.com/news/privacy/0,1848,52563,00.html.

[16] B. Scholkopf, A. Smola, K.-R. Muller "Nonlinear Component Analysis as a Kernel Eigenvalue Problem" *Neural Computation*, Vol. 10, pp. 1299-1319, 1998.

[17] C. Williams, M. Seeger "Using the Nyström Method to Speed up Kernel Machines," *Advances in Neural Information Processing Systems*, Vol 13, pp. 682-688, 2001.
